# Learning the Architecture of Sum-Product Networks Using Clustering on Variables

**Aaron Dennis**
Department of Computer Science
Brigham Young University
Provo, UT 84602
adennis@byu.edu

**Dan Ventura**
Department of Computer Science
Brigham Young University
Provo, UT 84602
ventura@cs.byu.edu

## Abstract

The sum-product network (SPN) is a recently-proposed deep model consisting of a network of sum and product nodes, and has been shown to be competitive with state-of-the-art deep models on certain difficult tasks such as image completion. Designing an SPN network architecture that is suitable for the task at hand is an open question. We propose an algorithm for learning the SPN architecture from data. The idea is to cluster variables (as opposed to data instances) in order to identify variable subsets that strongly interact with one another. Nodes in the SPN network are then allocated towards explaining these interactions. Experimental evidence shows that learning the SPN architecture significantly improves its performance compared to using a previously-proposed static architecture.

## 1 Introduction

The number of parameters in a textbook probabilistic graphical model (PGM) is an exponential function of the number of parents of the nodes in the graph. Latent variables can often be introduced such that the number of parents is reduced while still allowing the probability distribution to be represented. Figure 1 shows an example of modeling the relationship between symptoms of a set of diseases. The PGM at the left has no latent variables and the PGM at the right has an appropriately added "disease" variable. The model is able to be simplified because the symptoms are statistically independent of one another given the disease. The middle PGM shows a model in which the latent variable is introduced to no simplifying effect, demonstrating the need to be intelligent about what latent variables are added and how they are added.

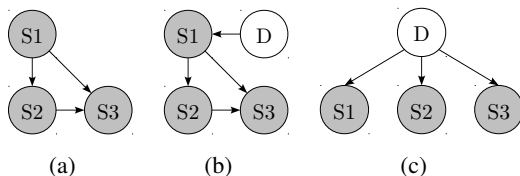

(a)        (b)        (c)

Figure 1: Introducing a latent variable. The PGM in (a) has no latent variables. The PGM in (b) has a latent variable introduced to no beneficial effect. The PGM in (c) has a latent variable that simplifies the model.

Deep models can be interpreted as PGMs that introduce multiple layers of latent variables over a layer of observed variables [1]. The architecture of these latent variables (the size of the layers, the number of variables, the connections between variables) can dramatically affect the performance of these models. Selecting a reasonable architecture is often done by hand.

This paper proposes an algorithm that automatically learns a deep architecture from data for a sum-product network (SPN), a recently-proposed deep model that takes advantage of the simplifying effect of latent variables [2]. Learning the appropri-

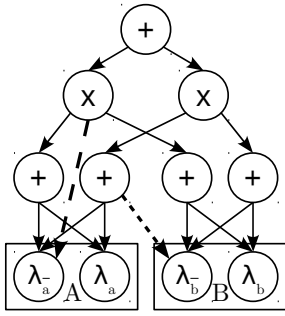

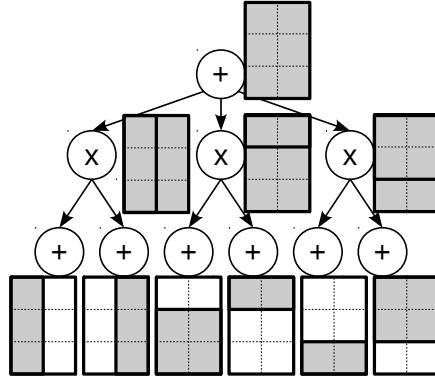

Figure 2: A simple SPN over two binary variables $A$ and $B$. The leaf node $\lambda_{\overline{a}}$ takes value 1 if $A = 0$ and 0 otherwise while leaf node $\lambda_a$ takes value 1 if $A = 1$ and 0 otherwise. If the value of $A$ is not known then both leaf nodes take value 1. Leaf nodes $\lambda_{\overline{b}}$ and $\lambda_b$ behave similarly. Weights on the edges connecting sum nodes with their children are not shown. The short-dashed edge causes the SPN to be incomplete. The long-dashed edge causes the SPN to be inconsistent.

Figure 3: The Poon architecture with $m = 1$ sum nodes per region. Three product nodes are introduced because the $2 \times 3$-pixel image patch can be split vertically and horizontally in three different ways. In general the Poon architecture has number-of-splits times $m^2$ product nodes per region.

ate architecture for a traditional deep model can be challenging [3, 4], but the nature of SPNs lend themselves to a remarkably simple, fast, and effective architecture-learning algorithm.

In proposing SPNs, Poon & Domingos introduce a general scheme for building an initial SPN architecture; the experiments they run all use one particular instantiation of this scheme to build an initial "fixed" architecture that is suitable for image data. We will refer to this architecture as the Poon architecture. Training is done by learning the parameters of an initial SPN; after training is complete, parts of the SPN may be pruned to produce a final SPN architecture. In this way both the weights and architecture are learned from data.

We take this a step further by also learning the initial SPN architecture from data. Our algorithm works by finding subsets of variables (and sets of subsets of variables) that are highly dependent and then effectively combining these together under a set of latent variables. This encourages the latent variables to act as mediators between the variables, capturing and representing the dependencies between them. Our experiments show that learning the initial SPN architecture in this way improves its performance.

## 2 Sum-Product Networks

Sum-product networks are rooted, directed acyclic graphs (DAGs) of sum, product, and leaf nodes. Edges connecting sum nodes to their children are weighted using non-negative weights. The value of a sum node is computed as the dot product of its weights with the values of it child nodes. The value of a product node is computed by multiplying the values of its child nodes. A simple SPN is shown in Figure 2.

Leaf node values are determined by the input to the SPN. Each input variable has an associated set of leaf nodes, one for each value the variable can take. For example, a binary variable would have two associated leaf nodes. The leaf nodes act as indicator functions, taking the value 1 when the variable takes on the value that the leaf node is responsible for and 0 otherwise.

An SPN can be constructed such that it is a representation of some probability distribution, with the value of its root node and certain partial derivatives with respect to the root node having probabilistic meaning. In particular, all marginal probabilities and many conditional probabilities can be computed [5]. Consequently an SPN can perform exact inference and does so efficiently when the size of the SPN is polynomial in the number of variables.

If an SPN does represent a probability distribution then we call it a valid SPN; of course, not all SPNs are valid, nor do they all facilitate efficient, exact inference. However, Poon & Domingos proved that if the architecture of an SPN follows two simple rules then it *will* be valid. (Note that this relationship does not go both ways; an SPN may be valid and violate one or both of these rules.) This, along with showing that SPNs can represent a broader class of distributions than other models that allow for efficient and exact inference are the key contributions made by Poon & Domingos.

To understand these rules it will help to know what the "scope of an SPN node" means. The scope of an SPN node $n$ is a subset of the input variables. This subset can be determined by looking at the leaf nodes of the subgraph rooted at $n$. All input variables that have one or more of their associated leaf nodes in this subgraph are included in the scope of the node. We will denote the scope of $n$ as $scope(n)$.

The first rule is that all children of a sum node must have the same scope. Such an SPN is called *complete*. The second rule is that for every pair of children, $(c_i, c_j)$, of a product node, there must not be contradictory leaf nodes in the subgraphs rooted at $c_i$ and $c_j$. For example, if the leaf node corresponding to the variable $X$ taking on value $x$ is in the subgraph rooted at $c_i$, then the leaf nodes corresponding to the variable $X$ taking on any other value may not appear in the subgraph rooted at $c_j$. An SPN following this rule is called *consistent*. The SPN in Figure 2 violates completeness (due to the short-dashed arrow) and it violates consistency (due to the long-dashed arrow).

An SPN may also be *decomposable*, which is a property similar to, but somewhat more restrictive than consistency. A decomposable SPN is one in which the scopes of the children of each product node are disjoint. All of the architectures described in this paper are decomposable.

Very deep SPNs can be built using these rules as a guide. The number of layers in an SPN can be on the order of tens of layers, whereas the typical deep model has three to five layers. Recently it was shown that deep SPNs can compute some functions using exponentially fewer resources than shallow SPNs would need [6].

The Poon architecture is suited for modeling probability distributions over images, or other domains with local dependencies among variables. It is constructed as follows. For every possible axis-aligned rectangular region in the image, the Poon architecture includes a set of $m$ sum nodes, all of whose scope is the set of variables associated with the pixels in that region. Each of these (non-single-pixel) regions are conceptually split vertically and horizontally in all possible ways to form pairs of rectangular subregions. For each pair of subregions, and for every possible pairing of sum nodes (one taken from each subregion), a product node is introduced and made the parent of the pair of sum nodes. The product node is also added as a child to all of the top region's sum nodes. Figure 3 shows a fragment of a Poon architecture SPN modeling a $2 \times 3$ image patch.

# 3   Cluster Architecture

As mentioned earlier, care needs to be taken when introducing latent variables into a model. Since the effect of a latent variable is to help explain the interactions between its child variables [7], it makes little sense to add a latent variable as the parent of two statistically independent variables.

In the example in Figure 4, variables $W$ and $X$ strongly interact and variables $Y$ and $Z$ do as well. But the relationship between all other pairs of variables is weak. The PGM in (a), therefore, allows latent variable $A$ to take account of the interaction between $W$ and $X$. On the other hand, variable $A$ does little in the PGM in (b) since $W$ and $Y$ are nearly independent. A similar argument can be made about variable $B$. Consequently, variable $C$ in the PGM in (a) can be used to explain the weak interactions between variables, whereas in the PGM in (b), variable $C$ essentially has the task of explaining the interaction between all the variables.

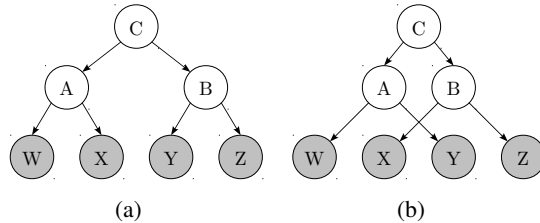

(a)                              (b)

Figure 4: Latent variables explain the interaction between child variables, causing the children to be independent given the latent variable parent. If variable pairs $(W, X)$ and $(Y, Z)$ strongly interact and other variable pairs do not, then the PGM in (a) is a more suitable model than the PGM in (b).

In the probabilistic interpretation of an SPN, sum nodes are associated with latent variables. (The evaluation of a sum node is equivalent to summing out its associated latent variable.) Each latent variable helps the SPN explain interactions between variables in the scope of the sum nodes. Just as in the example, then, we would like to place sum nodes over sets of variables with strong interactions.

The Poon architecture takes this principle into account. Images exhibit strong interactions between pixels in local spatial neighborhoods. Taking advantage of this prior knowledge, the Poon architecture chooses to place sum nodes over local spatial neighborhoods that are rectangular in shape.

There are a few potential problems with this approach, however. One is that the Poon architecture includes many rectangular regions that are long and skinny. This means that the pixels at each end of these regions are grouped together even though they probably have only weak interactions. Some grouping of weakly-interacting pixels is inevitable, but the Poon architecture probably does this more than is needed. Another problem is that the Poon architecture has no way of explaining strongly-interacting, non-rectangular local spatial regions. This is a major problem because such regions are very common in images. Additionally, if the data does not exhibit strong spatially-local interactions then the Poon architecture could perform poorly.

Our proposed architecture (we will call it the cluster architecture) avoids these problems. Large regions containing non-interacting pixels are avoided. Sum nodes can be placed over spatially-local, non-rectangular regions; we are not restricted to rectangular regions, but can explain arbitrarily-shaped blob-like regions. In fact, the regions found by the cluster architecture are not required to exhibit spatial locality. This makes our architecture suitable for modeling data that does not exhibit strong spatially-local interactions between variables.

## 3.1   Building a Cluster Architecture

As was described earlier, a sum node $s$ in an SPN has the task of explaining the interactions between all the variables in its scope. Let $scope(s) = \{V_1, \cdots, V_n\}$. If $n$ is large, then this task will likely be very difficult. SPNs have a mechanism for making it easier, however. Essentially, $s$ delegates part of its responsibilities to another set of sum nodes. This is done by first forming a partition of $scope(s)$, where $\{S_1, \cdots, S_k\}$ is a partition of $scope(s)$ if and only if $\bigcup_i S_i = scope(s)$ and $\forall i, j (S_i \cap S_j = \emptyset)$. Then, for each subset $S_i$ in the partition, an additional sum node $s_i$ is introduced into the SPN and is given the task of explaining the interactions between all the variables in $S_i$. The original sum node $s$ is then given a new child product node $p$ and the product node becomes the parent of each sum node $s_i$.

In this example the node $s$ is analogous to the variable $C$ in Figure 4 and the nodes $s_i$ are analogous to the variables $A$ and $B$. So this partitioning process allows $s$ to focus on explaining the interactions between the nodes $s_i$ and frees it from needing to explain everything about the interactions between the variables $\{V_1, \cdots, V_n\}$. And, of course, the partitioning process can be repeated recursively, with any of the nodes $s_i$ taking the place of $s$.

This is the main idea behind the algorithm for building a cluster architecture (see Algorithm 1 and Algorithm 2). However, due to the architectural flexibility of an SPN, discussing this algorithm in terms of sum and product nodes quickly becomes tedious and confusing. The following definition will help in this regard.

**Definition 1.** A **region graph** is a rooted DAG consisting of region nodes and partition nodes. The root node is a region node. Partition nodes are restricted to being the children of region nodes and vice versa. Region and partition nodes have scopes just like nodes in an SPN. The scope of a node $n$ in a region graph is denoted $scope(n)$.

Region nodes can be thought of as playing the role of sum nodes (explaining interactions among variables) and partition nodes can be thought of as playing the role of product nodes (delegating responsibilities). Using the definition of the region graph may not appear to have made things any simpler, but its benefits will become more clear when discussing the conversion of region graphs to SPNs (see Figure 5).

At a high level the algorithm for building a cluster architecture is simple: build a region graph (Algorithm 1 and Algorithm 2), then convert it to an SPN (Algorithm 3). These steps are described below.

**Algorithm 1** BuildRegionGraph

1: **Input:** training data $D$
2: $C' \leftarrow \text{Cluster}(D, 1)$
3: **for** $k = 2$ **to** $\infty$ **do**
4:     $C \leftarrow \text{Cluster}(D, k)$
5:     $r \leftarrow \text{Quality}(C)/\text{Quality}(C')$
6:     **if** $r < 1 + \delta$ **then**
7:         **break**
8:     **else**
9:         $C' \leftarrow C$
10: $G \leftarrow \text{CreateRegionGraph}()$
11: $n \leftarrow \text{AddRegionNodeTo}(G)$
12: **for** $i = 1$ **to** $k$ **do**
13:     $\text{ExpandRegionGraph}(G, n, C_i)$

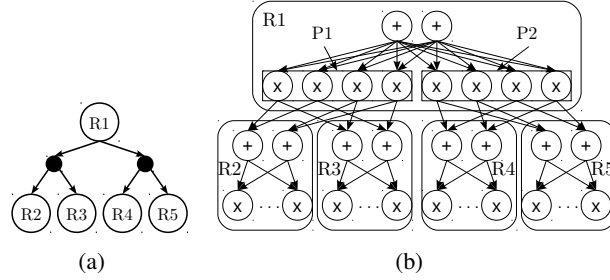

(a)                        (b)

Figure 5: Subfigure (a) shows a region graph fragment consisting of region nodes $R_1$, $R_2$, $R_3$, $R_4$, and $R_5$. $R_1$ has two parition nodes (the smaller, filled-in nodes). Subfigure (b) shows the region graph converted to an SPN. In the SPN each region is allotted two sum nodes. The product nodes in $R_1$ are surrounded by two rectangles labeled $P_1$ and $P_2$; they correspond to the partition nodes in the region graph.

Algorithm 1 builds a region graph using training data to guide the construction. In lines 2 through 9 the algorithm clusters the training instances into $k$ clusters $C = \{C_1, \cdots, C_k\}$. Our implementation uses the scikit-learn [8] implementation of $k$-means to cluster the data instances, but any clustering method could be used. The value for $k$ is chosen automatically; larger values of $k$ are tried until increasing the value does not substantially improve a cluster-quality score. The remainder of the algorithm creates a single-node region graph $G$ and then adds nodes and edges to $G$ using $k$ calls to Algorithm 2 (ExpandRegionGraph). To encourage the expansion of $G$ in different ways, a different subset of the training data, $C_i$, is passed to ExpandRegionGraph on each call.

At a high level, Algorithm 2 partitions scopes into sub-scopes recursively, adding region and partition nodes to $G$ along the way. The initial call to ExpandRegionGraph partitions the scope of the root region node. A corresponding partition node is added as a child of the root node. Two sub-region nodes (whose scopes form the partition) are then added as children to the partition node. Algorithm 2 is then called recursively with each of these sub-region nodes as arguments (unless the scope of the sub-region node is too small).

In line 3 of Algorithm 2 the PartitionScope function in our implementation uses the $k$-means algorithm in an unusual way. Instead of partitioning the instances of the training dataset $D$ into $k$ instance-clusters, it partitions variables into $k$ variable-clusters as follows. $D$ is encoded as a matrix, each row being a data instance and each column corresponding to a variable. Then $k$-means is run on $D^T$, causing it to partition the variables into $k$ clusters. Actually, the PartitionScope function is only supposed to partition the variables in $scope(n)$, not all the variables (note its input parameter). So before calling $k$-means we build a new matrix $D_n$ by removing columns from $D$, keeping only those columns that correspond to variables in $scope(n)$. *Then $k$-means is run on $D_n^T$ and the resulting variable partition is returned.* The $k$-means algorithm serves the purpose of detecting subsets of variables that strongly interact with one another. Other methods (including other clustering algorithms) could be used in its place.

After the scope $S_n$ of a node $n$ has been partitioned into $S_1$ and $S_2$, Algorithm 2 (lines 4 through 11) looks for region nodes in $G$ whose scope is similar to $S_1$ or $S_2$; if region node $r$ with scope $S_r$ is such a node, then $S_1$ and $S_2$ are adjusted so that $S_1 = S_r$ and $\{S_1, S_2\}$ is still a partition of $S_n$. Lines 12 through 18 expand the region graph based on the partition of $S_n$. If node $n$ does not already have a child partition node representing the partition $\{S_1, S_2\}$ then one is created ($p$ in line 15); $p$ is then connected to child region nodes $n_1$ and $n_2$, whose scopes are $S_1$ and $S_2$, respectively.

Note that $n_1$ and $n_2$ may be newly-created region nodes or they may be nodes that were created during a previous call to Algorithm 2. We recursively call ExpandRegionGraph only on newly-created nodes; the recursive call is also not made if the node is a leaf node ($|S_i| = 1$) since partitioning a leaf node is not helpful (see lines 19 through 22).

**Algorithm 2** ExpandRegionGraph

1: **Input:** region graph $G$,
      region node $n$ in $G$, training data $D$
2: $S_n \leftarrow scope(n)$
3: $\{S_1, S_2\} \leftarrow$ PartitionScope$(S_n, D)$
4: $S \leftarrow$ ScopesOfAllRegionNodesIn$(G)$
5: **for all** $S_r \in S$ s.t. $S_r \subset S_n$ **do**
6:    $p_1 \leftarrow |S_1 \cap S_r|/|S_1 \cup S_r|$
7:    $p_2 \leftarrow |S_2 \cap S_r|/|S_2 \cup S_r|$
8:    **if** $\max\{p_1, p_2\} > threshold$ **then**
9:       $S_1 \leftarrow S_r$
10:      $S_2 \leftarrow S_n \setminus S_r$
11:      **break**
12: $n_1 \leftarrow$ GetOrCreateRegionNode$(G, S_1)$
13: $n_2 \leftarrow$ GetOrCreateRegionNode$(G, S_2)$
14: **if** PartitionDoesNotExist$(G, n, n_1, n_2)$ **then**
15:    $p \leftarrow$ NewPartitionNode()
16:    AddChildToRegionNode$(n, p)$
17:    AddChildToPartitionNode$(p, n_1)$
18:    AddChildToPartitionNode$(p, n_2)$
19: **if** $S_1 \notin S \wedge |S_1| > 1$ **then**
20:    ExpandRegionGraph$(G, n_1)$
21: **if** $S_2 \notin S \wedge |S_2| > 1$ **then**
22:    ExpandRegionGraph$(G, n_2)$

**Algorithm 3** BuildSPN

**Input:** region graph $G$, sums per region $m$
**Output:** SPN $S$
$R \leftarrow$ RegionNodesIn$(G)$
**for all** $r \in R$ **do**
  **if** IsRootNode$(r)$ **then**
    $N \leftarrow$ AddSumNodesToSPN$(S, 1)$
  **else**
    $N \leftarrow$ AddSumNodesToSPN$(S, m)$
  $P \leftarrow$ ChildPartitionNodesOf$(r)$
  **for all** $p \in P$ **do**
    $C \leftarrow$ ChildrenOf$(p)$
    $O \leftarrow$ AddProductNodesToSPN$(S, m^{|C|})$
    **for all** $n \in N$ **do**
      AddChildrenToSumNode$(n, O)$
    $Q \leftarrow$ empty list
    **for all** $c \in C$ **do**
      //We assume the sum nodes associated
      //with $c$ have already been created.
      $U \leftarrow$ SumNodesAssociatedWith$(c)$
      AppendToList$(Q, U)$
    ConnectProductsToSums$(O, Q)$
**return** $S$

After the $k$ calls to Algorithm 2 have been made, the resulting region graph must be converted to an SPN. Figure 5 shows a small subgraph from a region graph and its conversion into an SPN; this example demonstrates the basic pattern that can be applied to all region nodes in $G$ in order to generate an SPN. A more precise description of this conversion is given in Algorithm 3. In this algorithm the assumption is made (noted in the comments) that certain sum nodes are inserted before others. This assumption can be guaranteed if the algorithm performs a postorder traversal of the region nodes in $G$ in the outermost loop. Also note that the ConnectProductsToSums method connects product nodes of the current region with sum nodes from its subregions; the children of a product node consist of a single node drawn from each subregion, and there is a product node for every possible combination of such sum nodes.

# 4  Experiments and Results

Poon showed that SPNs can outperform deep belief networks (DBNs), deep Boltzman machines (DBMs), principle component analysis (PCA), and a nearest-neighbors algorithm (NN) on a difficult image completion task. The task is the following: given the right/top half of an image, paint in the left/bottom half of it. The completion results of these models were compared qualitatively by inspection and quantitatively using mean squared error (MSE). SPNs produced the best results; our experiments show that the cluster architecture significantly improves SPN performance.

We matched the experimental set-up reported in [2] in order to isolate the effect of changing the initial SPN architecture and to make their reported results directly comparable to several of our results. They add 20 sum nodes for each non-unit and non-root region. The root region has one sum node and the unit regions have four sum nodes, each of which function as a Gaussian over pixel values. The Gaussians means are calculated using the training data for each pixel, with one Gaussian covering each quartile of the pixel-values histogram. Each training image is normalized such that its mean pixel value is zero with a standard deviation of one. Hard expectation maximization (EM) is used to train the SPNs; mini-batches of 50 training instances are used to calculate each weight update. All sum node weights are initialized to zero; weight values are decreased after each training epoch using an $L_0$ prior; add-one smoothing on sum node weights is used during network evaluation.

Table 1: Results of experiments on the Olivetti, Caltech 101 Faces, artificial, and shuffled-Olivetti datasets comparing the Poon and cluster architectures. Negative log-likelihood (LLH) of the training set and test set is reported along with the MSE for the image completion results (both left-half and bottom-half completion results).

| Dataset | Measurement | Poon | Cluster |
|---|---|---|---|
| Olivetti | Train LLH | $318 \pm 1$ | $433 \pm 17$ |
| | Test LLH | $863 \pm 9$ | $715 \pm 31$ |
| | MSE (left) | $996 \pm 42$ | $814 \pm 35$ |
| | MSE (bottom) | $963 \pm 42$ | $820 \pm 38$ |
| Caltech | Train LLH | $289 \pm 4$ | $379 \pm 8$ |
| Faces | Test LLH | $674 \pm 15$ | $557 \pm 11$ |
| | MSE (left) | $1968 \pm 89$ | $1746 \pm 87$ |
| | MSE (bottom) | $1925 \pm 82$ | $1561 \pm 44$ |
| Artificial | Train LLH | $195 \pm 0$ | $169 \pm 0$ |
| | Test LLH | $266 \pm 4$ | $223 \pm 6$ |
| | MSE (left) | $842 \pm 51$ | $558 \pm 27$ |
| | MSE (bottom) | $877 \pm 85$ | $561 \pm 29$ |
| Shuffled | Train LLH | $793 \pm 3$ | $442 \pm 14$ |
| | Test LLH | $1193 \pm 3$ | $703 \pm 14$ |
| | MSE (left) | $811 \pm 11$ | $402 \pm 16$ |
| | MSE (bottom) | $817 \pm 17$ | $403 \pm 17$ |

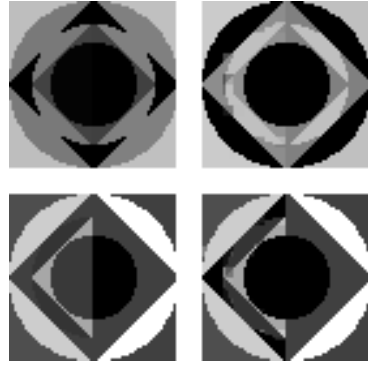

Figure 6: A cluster-architecture SPN completed the images in the left column and a Poon-architecture SPN completed the images in the right column. All images shown are left-half completions. The top row is the best results as measured by MSE and the bottom row is the worst results. Note the smooth edges in the cluster completions and the jagged edges in the Poon completions.

We test the cluster and Poon architectures by learning on the Olivetti dataset [9], the faces from the Caltech-101 dataset [10], an artificial dataset that we generated, and the shuffled-Olivetti dataset, which the Olivetti dataset with the pixels randomly shuffled (all images are shuffled in the same way). The Caltech-101 faces were preprocessed as described by Poon & Domingos. The cluster architecture is compared to the Poon architectures using the negative log-likelihood (LLH) of the training and test sets as well as the MSE of the image completion results for the left half and bottom half of the images. We train ten cluster architecture SPNs and ten Poon architecture SPNs. Average results across the ten SPNs along with the standard deviation are given for each measurement.

On the Olivetti and Caltech-101 Faces datasets the Poon architecture resulted in better training set LLH, but the cluster architecture generalized better, getting a better test set LLH (see Table 1). The cluster architecture was also clearly better at the image completion tasks as measured by MSE.

The difference between the two architectures is most pronounced on the artificial dataset. The images in this dataset are created by pasting randomly-shaded circle- and diamond-shaped image patches on top of one another (see Figure 6), ensuring that various pixel patches are statistically independent. The cluster architecture outperforms the Poon architecture across all measures on this dataset (see Table 1); this is due to its ability to focus resources on non-rectangular regions.

To demonstrate that the cluster architecture does not rely on the presence of spatially-local, strong interactions between the variables, we repeated the Olivetti experiment with the pixels in the images having been shuffled. In this experiment (see Table 1) the cluster architecture was, as expected, relatively unaffected by the pixel shuffling. The LLH measures remained basically unchanged from the Olivetti to the Olivetti-shuffled datasets. (The MSE results did not stay the same because the image completions happened over different subsets of the pixels.) On the other hand, the performance of the Poon architecture dropped considerably due to the fact that it was no longer able to take advantage of strong correlations between neighboring pixels.

Figure 7 visually demonstrates the difference between the rectangular-regions Poon architecture and the arbitrarily-shaped-regions cluster architecture. Artifacts of the different region shapes can be seen in subfigure (a), where some regions are shaded lighter or darker, revealing region boundaries. Subfigure (b) compares the best of both architectures, showing image completion results on which both architectures did well, qualitatively speaking. Note how the Poon architecture produces results that look "blocky", whereas the cluster architecture produces results that are smoother-looking.

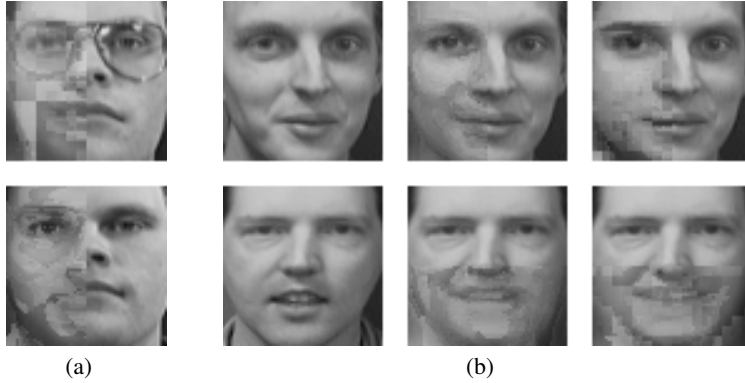

<div align="center">(a)                     (b)</div>

Figure 7: The completion results in subfigure (a) highlight the difference between the rectangular-shaped regions of the Poon architecture (top image) and the blob-like regions of the cluster architecture (bottom image), artifacts of which can be seen in the completions. Subfigure (b) shows ground truth images, cluster-architecture SPN completions, and Poon-architecture SPN completions in the left, middle, and right columns respectively. Left-half completions are in the top row and bottom-half completions are in the bottom row.

Table 2: Test set LLH values for the Olivetti, Olivetti45, and Olivetti4590 datasets for different values of $k$. For each dataset the best LLH value is marked in bold.

| Dataset / $k$ | 1 | 2 | 3 | 4 | 5 | 6 | 7 | 8 |
|---|---|---|---|---|---|---|---|---|
| Olivetti | **650** | 653 | 671 | 685 | 711 | 716 | 717 | 741 |
| Olivetti45 | 523 | **495** | 508 | 529 | 541 | 528 | 544 | 532 |
| Olivetti4590 | 579 | 576 | **550** | 554 | 577 | 595 | 608 | 592 |

Algorithm 1 expands a region graph $k$ times (lines 12 and 13). The value of $k$ can significantly affect test set LLH, as shown in Table 2. A value that is too low leads to an insufficiently powerful model and a value that is too high leads to a model that overfits the training data and generalizes poorly.

A singly-expanded model ($k = 1$) is optimal for the Olivetti dataset. This may be due in part to the Olivetti dataset having only one distinct class of images (faces in a particular pose). Datasets with more image classes may benefit from additional expansions. To experiment with this hypothesis we create two new datasets: Olivetti45 and Olivetti4590. Olivetti45 is created by augmenting the Olivetti dataset with Olivetti images that are rotated by $-45$ degrees. Olivetti4590 is built similarly but with rotations by $-45$ degrees and by $-90$ degrees. The Olivetti45 dataset, then, has two distinct classes of images: rotated and non-rotated. Similarly, Olivetti4590 has three distinct image classes. Table 2 shows that, as expected, the optimal value of $k$ for the Olivetti45 and Olivetti4590 datasets is two and three, respectively.

Note that the Olivetti test set LLH with $k = 1$ in Table 2 is better than the test set LLH reported in Table 1. This shows that the algorithm for automatically selecting $k$ in Algorithm 1 is not optimal. Another option is to use a hold-out set to select $k$, although this method may not not be appropriate for small datasets.

## 5   Conclusion

The algorithm for learning a cluster architecture is simple, fast, and effective. It allows the SPN to focus its resources on explaining the interactions between arbitrary subsets of input variables. And, being driven by data, the algorithm guides the allocation of SPN resources such that it is able to model the data more efficiently. Future work includes experimenting with alternative clustering algorithms, experimenting with methods for selecting the value of $k$, and experimenting with variations of Algorithm 2 such as generalizing it to handle partitions of size greater than two.

# References

[1] Geoffrey E. Hinton, Simon Osindero, and Yee-Whye Teh. A fast learning algorithm for deep belief nets. *Neural Computation*, 18:1527–1554, July 2006.

[2] Hoifung Poon and Pedro Domingos. Sum-product networks: A new deep architecture. In *Proceedings of the Twenty-Seventh Annual Conference on Uncertainty in Artificial Intelligence (UAI-11)*, pages 337–346, Corvallis, Oregon, 2011. AUAI Press.

[3] Ryan Prescott Adams, Hanna M. Wallach, and Zoubin Ghahramani. Learning the structure of deep sparse graphical models. In *Proceedings of the 13th International Conference on Artificial Intelligence and Statistics*, 2010.

[4] Nevin L. Zhang. Hierarchical latent class models for cluster analysis. *Journal of Machine Learning Research*, 5:697–723, December 2004.

[5] Adnan Darwiche. A differential approach to inference in bayesian networks. *Journal of the ACM*, 50:280–305, May 2003.

[6] Olivier Delalleau and Yoshua Bengio. Shallow vs. deep sum-product networks. In *Advances in Neural Information Processing Systems 24*, pages 666–674. 2011.

[7] Daphne Koller and Nir Friedman. *Probabilistic Graphical Models: Principles and Techniques*. MIT Press, 2009.

[8] F. Pedregosa, G. Varoquaux, A. Gramfort, V. Michel, B. Thirion, O. Grisel, M. Blondel, P. Prettenhofer, R. Weiss, V. Dubourg, J. Vanderplas, A. Passos, D. Cournapeau, M. Brucher, M. Perrot, and E. Duchesnay. Scikit-learn: Machine learning in python. *Journal of Machine Learning Research*, 12:2825–2830, 2011.

[9] F.S. Samaria and A.C. Harter. Parameterisation of a stochastic model for human face identification. In *Proceedings of the Second IEEE Workshop on Applications of Computer Vision*, pages 138 –142, Dec 1994.

[10] Li Fei-Fei, R. Fergus, and P. Perona. Learning generative visual models from few training examples: An incremental bayesian approach tested on 101 object categories. In *IEEE CVPR 2004, Workshop on Generative-Model Based Vision*, 2004.

